# Adaptation in Speech Motor Control

**John F. Houde***
UCSF Keck Center
Box 0732
San Francisco, CA 94143
houde@phy.ucsf.edu

**Michael I. Jordan**
MIT Dept. of Brain and Cognitive Sci.
E10-034D
Cambridge, MA 02139
jordan@psyche.mit.edu

## Abstract

Human subjects are known to adapt their motor behavior to a shift of the visual field brought about by wearing prism glasses over their eyes. We have studied the analog of this effect in speech. Using a device that can feed back transformed speech signals in real time, we exposed subjects to alterations of their own speech feedback. We found that speakers learn to adjust their production of a vowel to compensate for feedback alterations that change the vowel's perceived phonetic identity; moreover, the effect generalizes across consonant contexts and to different vowels.

## 1  INTRODUCTION

For more than a century, it has been know that humans will adapt their reaches to altered visual feedback [8]. One of the most studied examples of this adaptation is prism adaptation, which is seen when a subject reaches to targets while wearing image-shifting prism glasses [2]. Initially, the subject misses the targets, but he soon learns to compensate and reach accurately. This compensation is retained beyond the time that the glasses are worn: when the glasses are removed, the subject's reaches now overshoot targets in the direction that he compensated. This retained compensation is called adaptation, and its generation from exposure to altered sensory feedback is called *sensorimotor adaptation* (SA).

In the study reported here, we investigated whether SA could be observed in a motor task that is quite different from reaching – speech production. Specifically, we examined whether the control of phonetically relevant speech features would respond adaptively to altered auditory feedback. By itself, this is an important theoretical question because various aspects of speech production have already been shown to be sensitive to auditory feedback [5, 1, 4]. Moreover, we were particularly

interested in whether speech SA would also exhibit generalization. If so, speech SA could be used to examine the organization of speech motor control. For example, suppose we observed adaptation of [ε] in "get". We could then examine whether we also see adaptation of [ε] in "peg". If so, then producing [ε] in the two different words must access a common, adapted representation – evidence for a hierarchical speech production system in which word productions are composed from smaller units such as phonemes. We could also examine whether adapting [ε] in "get" causes adaptation of [æ] in "gat". If so, then the production representations of [ε] and [æ] could not be independent, supporting the idea that vowels are produced by controlling a common set of features. Such theories about the organization of the speech production system have been postulated in phonology and phonetics, but the empirical evidence supporting these theories has generally been observational and hence not entirely conclusive [7, 6].

## 2 METHODS

To study speech SA, we focused on vowel production because the phonetically relevant features of vowel sounds are formant frequencies, which are feasible to alter in real time.[1]

To alter the formants of a subject's speech feedback, we built the apparatus shown in Figure 1. The subject wears earphones and a microphone and sits in front of a PC video monitor that presents words to be spoken aloud. The signal from the microphone is sent to a Digital Signal Processing board, which collects a 64ms time interval from which a magnitude spectrum is calculated. From this spectrum, formant frequencies and amplitudes are estimated. To alter the speech, the first three formant frequencies are shifted, and the shifted formants drive a formant synthesizer that creates the output speech sent to the subject's earphones. This analysis-synthesis process was accomplished with only 16ms of feedback delay. To minimize how much the subject directly heard of his own voice via bone conduction, the subject produced only whispered speech, masked with mild noise.

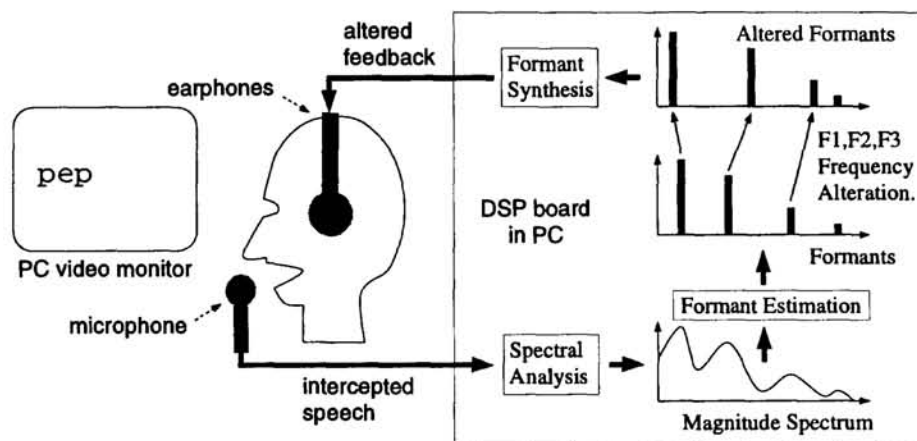

Figure 1: The apparatus used in the study.

For each subject in our experiment, we shifted formants along the path defined by the (F1,F2,F3) frequencies of a subject's productions of the vowels [i], [ɩ], [ε], [æ],

and [a].[2] Figure 2 shows examples of this shifting process in (F1,F2) space for the feedback transformations that were used in the study. To shift formants along the subject's [i]-[a] path, we extend the path at both ends and we number the endpoints and vowels to make a path position measure that normalizes the distances between vowels. The formants of each speech sound F produced by the subject were then re-represented in terms of path projection – the path position of nearest path point P, and path deviation – the distance D to this point P. Feedback transformations were constructed to alter path projections while preserving path deviations. Two different transformations were used. The +2.0 transformation added 2.0 to path projections: under this transform, if the subject produced speech sound F (a sound near [ε]), he heard instead sound F+ (a sound near [a]). The subject could compensate for this transform and hear sound F only by shifting his production of F to F- (a sound near [i]). The -2.0 transformation subtracted 2.0 from path projections: under this transformation, if the subject produced F, he heard F-. Thus, in this case, the subject could compensate by shifting production to F+.

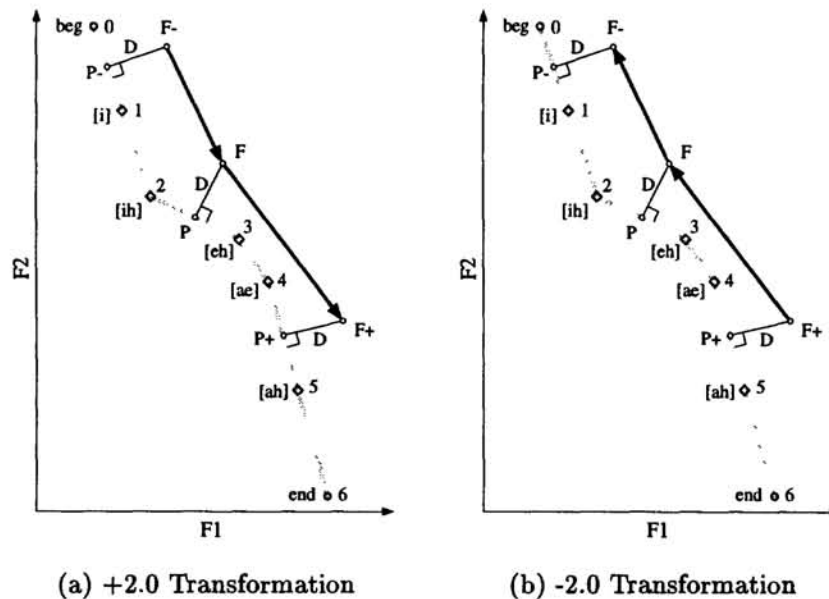

| (a) +2.0 Transformation | (b) -2.0 Transformation |

Figure 2: Feedback transformations used in the study.

These feedback transformations were used in an experiment in which a subject was visually prompted to whisper words with a 300ms target duration. Word promptings occurred in groups of ten called *epochs*. Within each epoch, the first six words came from a set of training words and the last four came from a set of testing words. The subject heard feedback of his first five word productions in each epoch, while masking noise blocked his hearing for his remaining five word productions in the epoch. Thus, the subject only heard feedback of his production of the first five training words and never heard his productions of the testing words.

The experiment lasted 2 hours and consisted of 422 epochs divided over five phases:

1. A 10 minute *warmup phase* used to acclimate the subject to the experimental setup.

2. A 17 minute *baseline phase* used to measure formants of the subject's normal vowel productions.

3. A 20 minute *ramp phase* in which the subject's feedback was increasingly altered up to a maximum value.

4. A 1 hour *training phase* in which the subject produced words while the feedback was maximally altered.

5. A 17 minute *test phase* used to measure formants of the subject's post-exposure vowel productions while his feedback was maximally altered.

By the end of the ramp phase, feedback alteration reached its maximum strength, which was +2.0 for half the subjects and -2.0 for the other subjects. In addition, all subjects were run in a control experiment in which feedback was never altered.

The two word sets from which prompted words were selected were both sets of CVC words. Training words (in which adaptation was induced) were all bilabials with [ɛ] as the vowel ("pep", "peb", "bep", and "beb"). Testing words (in which generalization of the training word adaptation was measured) were divided into two subsets, each designed to measure a different type of generalization: (1) context generalization words, which had the same vowel [ɛ] as the training words but varied the consonant context ("peg", "gep", and "teg"); (2) vowel target generalization words, which had the same consonant context as the training words but varied the vowel ("pip,", "peep,","pap", and "pop").

Eight male MIT students participated in the study. All were native speakers of North American English and all were naive to the purpose of the study.

## 3 RESULTS

To illustrate how we measured compensation and adaptation in the experiments, we first show the results for an individual subject. Figure 3 shows (F1,F2) plots of response of subject OB in both the adaptation experiment (in which he was exposed to the -2.0 feedback transformation) and the control experiment. In each figure, the dotted line is OB's [i]-[a] path.

Figure 3(a) shows OB's compensation responses, which were measured from his productions of the training words made when he heard feedback of his whispering. The solid arrow labeled "-2.0 xform" shows how much his mean vowel formants changed (testing phase - baseline phase) after being exposed to the -2.0 feedback transformation. It shows he shifted his production of [ɛ] to something a bit past [æ], which corresponds to a path projection change of slightly more than one vowel interval towards [a]. Thus, since the path projection shift of the transform was -2.0 (2.0 vowel intervals towards [i]), the figure shows that OB compensates for over half the action of the transformation. The hollow arrow in Figure 3(a) shows how OB heard his compensation. It shows he heard his actual production shift from [ɛ] towards [a] as a shift from [i] back towards [ɛ].

Figure 3(b) shows how much of OB's compensation was retained when he whispered the training words with feedback blocked by noise. This retained compensation is called adaptation, and it was measured from path projection changes by the same method used to measure compensation. In the figure, we see OB's adaptation

response (the solid "-2.0 xform" arrow) is a path projection shift of slightly less than one vowel interval, so his adaptation is slightly less than half. Thus, the figure shows that OB retains an appreciable amount of his compensation in the absence of feedback.

Finally, in both plots of Figure 3, the almost non-existent "control" arrows show that OB exhibited almost no formant change in the control experiment – as we would expect since feedback was never altered in this experiment.

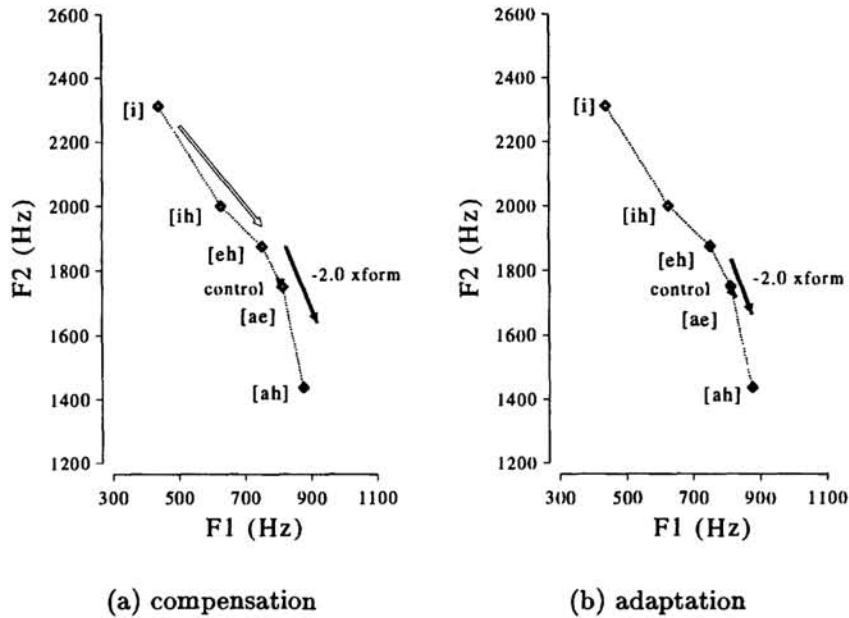

(a) compensation                              (b) adaptation

Figure 3: Subject OB compensation and adaptation.

The plots in Figure 4 show that there was significant compensation and adaptation across all subjects. In these plots, the vertical scale indicates how much the changes in mean vowel formants (testing phase - baseline phase) in each subject's productions of the training words compensated for the action of the feedback transformation he was exposed to. The filled circles linked by the solid line show compensation (Figure 4(a)) and retained compensation, or adaptation (Figure 4(b)) across subjects in the adaptation experiment in which feedback was altered; the open circles linked by the dotted line show the same measures from the control experiment in which feedback was not altered. (The solid and dotted lines facilitate comparison of results across subjects but do not signify any relationship between subjects.) In the control experiment, for each subject, compensation and adaptation were measured with respect to the feedback transformation used in the adaptation experiment.

The plots show that there are large variations in compensation and adaptation across subjects, but overall there was significantly more compensation ($p < 0.006$) and adaptation ($p < 0.023$) in the adaptation experiments that in the control experiments.

Figure 5 shows plots of how much of the adaptation observed in the training words carried over the the testing words. For each testing word shown, a measure of this carryover called mean generalization is plotted, which was calculated as a ratio of adaptations: the adaptation seen in the testing word divided by the adaptation seen in the training words (adaptation values observed in the control experiment were

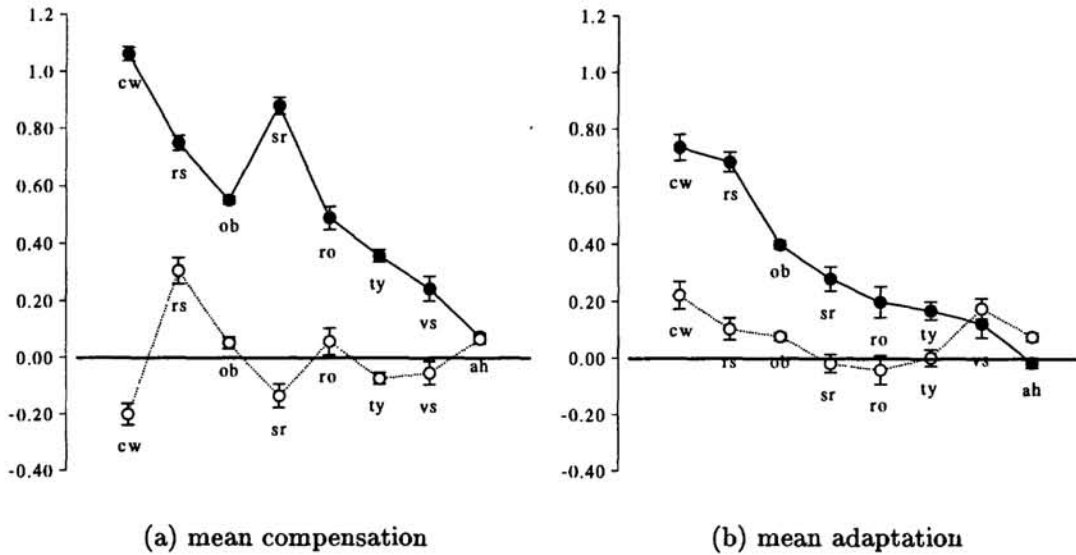

(a) mean compensation          (b) mean adaptation

Figure 4: Mean compensation and adaptation across all subjects.

subtracted out to remove any effects not arising from exposure to altered feedback).

Figure 5(a) shows mean generalization for the context generalization words except for "pep" (since "pep" was also a training word). The plot shows large variance in mean generalization for each of the three words, but overall there was significant ($p < 0.040$) mean generalization. Thus, there was significant carryover of the adaptation of [ε] in the training words to different consonant contexts.

Figure 5(b) shows mean generalization for the vowel target generalization words. Not all of these words are shown: unfortunately, we weren't able to accurately estimate the formants of [i] and [a], so "peep" and "pop" were dropped from our generalization analysis. For the remaining two vowel target generalization words, the plot shows large variance in mean generalization for each of the words, but overall there was significant ($p < 0.013$) mean generalization. Thus, there was significant generalization of the adaptation of [ε] to the vowels [ι] and [æ].

# 4   DISCUSSION

Several conclusions can be drawn from the experiment described above. First, comparison of the adaptation and control experiment results seen in Figure 4 shows a clear effect of exposure to the altered feedback: this exposure caused compensation responses in most subjects. Furthermore, the adaptation results show that this compensation was retained in absence of acoustic feedback. Next, the context generalization results seen in Figure 5(a) show that some adapted representation of [ε] is shared across the training and testing words. These results provide evidence for a hierarchical speech production system in which words are composed from smaller phoneme-like units. Finally, the vowel target generalization results seen in Figure 5(b) show that the production representations of [ι], [ε], and [æ] are not independent, suggesting that these vowels are produced by controlling a common set of features.

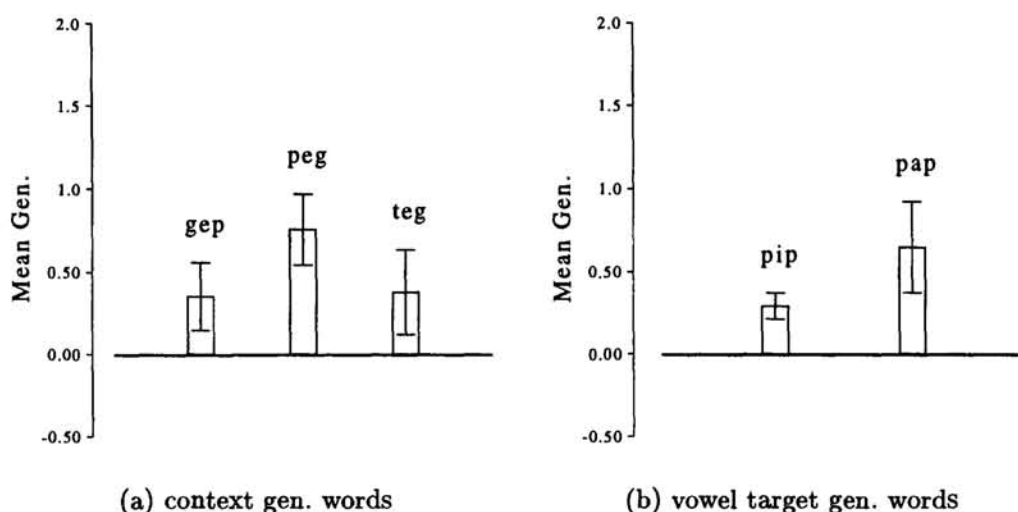

<center>(a) context gen. words                    (b) vowel target gen. words</center>

Figure 5: Mean generalization for the testing words, averaged across subjects.

Thus, in summary, our study has shown (1) that speech production, like reaching, can be made to exhibit sensorimotor adaptation, and (2) that this adaptation effect exhibits generalization that can be used to make inferences about the structure of the speech production system.

### Acknowledgments

We thank J. Perkell, K. Stevens, R. Held and P. Sabes for helpful discussions.

## Footnotes

*To whom correspondence should be addressed.

[1]See [3] for detailed discussion of the methods used in this study.

[2]Where possible, we use standard phonetic symbols for vowel sounds: [i] as in "seat", [ι] as in "hit", [ε] as in "get", [æ] as in "hat", and [a] as in "pop". Where font limitations prevent us from using these symbols, we use the alternate notation of [i], [ih], [eh], [ae], and [ah], respectively, for the same vowel sounds.

## References

[1] V. L. Gracco et al., (1994) *J. Acoust. Soc. Am.* 95:2821

[2] H. V. Helmholtz, (1867) *Treatise on physiological optics, Vol. 3* (Eng. Trans. by Optical Soc. of America, Rochester, NY, 1925)

[3] J. F. Houde (1997), *Sensorimotor Adaptation in Speech Production*, Doctoral Dissertation, M. I. T., Cambridge, MA.

[4] H. Kawahara (1993) *J. Acoust. Soc. Am.* 94:1883.

[5] B. S. Lee (1950) *J. Acoust. Soc. Am.* 22:639.

[6] W. J. M. Levelt (1989), *Speaking: from intention to articulation*, MIT Press, Cambridge, MA.

[7] A. S. Meyer (1992), *Cognition* 42:181.

[8] R. B. Welch (1986), *Handbook of Perception and Human Performance*, K. R. Boff, L. Kaufman, J. P. Thomas Eds., John Wiley and Sons, New York.